# An Improved Decomposition Algorithm for Regression Support Vector Machines

**Pavel Laskov**
Department of Computer and Information Sciences
University of Delaware
Newark, DE 19718
*laskov@asel.udel.edu*

## Abstract

A new decomposition algorithm for training regression Support Vector Machines (SVM) is presented. The algorithm builds on the basic principles of decomposition proposed by Osuna et. al., and addresses the issue of optimal working set selection. The new criteria for testing optimality of a working set are derived. Based on these criteria, the principle of "maximal inconsistency" is proposed to form (approximately) optimal working sets. Experimental results show superior performance of the new algorithm in comparison with traditional training of regression SVM without decomposition. Similar results have been previously reported on decomposition algorithms for pattern recognition SVM. The new algorithm is also applicable to advanced SVM formulations based on regression, such as density estimation and integral equation SVM.

## 1 Introduction

The increasing interest in applications of Support Vector Machines (SVM) to large-scale problems ushers in new requirements for computational complexity of their training algorithms. Requests have been recently made for algorithms capable of handling problems containing $10^5$ - $10^6$ examples [1]. Training an SVM constitutes a quadratic programming problem, and a typical SVM package uses an off-the-shelf optimization software to obtain a solution to it. The number of variables in the optimization problem is equal to the number of training data points (for the pattern recognition SVM) or twice that number (for the regression SVM). The speed of general-purpose optimization methods is insufficient for problems containing more than a few thousand examples. This has motivated a quest for special-purpose training algorithms to take advantage of the particular structure of SVM training problems.

The main avenue of research in SVM training algorithms is decomposition. The key idea of decomposition, due to Osuna et. al. [2], is to freeze all but a small number of optimization variables, and to solve a sequence of small fixed-size problems. The set of variables whose values are optimized at a current iteration is called the *working set*. Complexity of re-optimizing the working set is assumed to be constant-time.

In order for a decomposition algorithm to be successful, the working set must be selected in a smart way. The fastest known decomposition algorithm is due to Joachims [3]. It is based on Zoutendijk's method of feasible directions proposed in the optimization community in the early 1960's. However Joachims' algorithm is limited to pattern recognition SVM because it makes use of labels being $\pm 1$. The current article presents a similar algorithm for the regression SVM.

The new algorithm utilizes a slightly different background from optimization theory. The Karush-Kuhn-Tucker Theorem is used to derive conditions for determining whether or not a given working set is optimal. These conditions become the algorithm's termination criteria, as an alternative to Osuna's criteria (also used by Joachims without modification) which used conditions for individual points. The advantage of the new conditions is that knowledge of the hyperplane's constant factor $b$, which in some cases is difficult to compute, is not required. Further investigation of the new termination conditions allows to form the strategy for selecting an optimal working set. The new algorithm is applicable to the pattern recognition SVM, and is provably equivalent to Joachims' algorithm. One can also interpret the new algorithm in the sense of the method of feasible directions. Experimental results presented in the last section demonstrate superior performance of the new method in comparison with traditional training of regression SVM.

## 2    General Principles of Regression SVM Decomposition

The original decomposition algorithm proposed for the pattern recognition SVM in [2] has been extended to the regression SVM in [4]. For the sake of completeness I will repeat the main steps of this extension with the aim of providing terse and streamlined notation to lay the ground for working set selection.

Given the training data of size $l$, training of the regression SVM amounts to solving the following quadratic programming problem in $2l$ variables:

$$
\begin{aligned}
\text{Maximize} \quad W(\tilde{\alpha}) \quad &= \quad \bar{\mathbf{y}}^T \tilde{\alpha} - \frac{1}{2}\tilde{\alpha}^T D \tilde{\alpha} \\
\text{subject to:} \quad \mathbf{c}^T \tilde{\alpha} \quad &= \quad 0 \\
\tilde{\alpha} - C\mathbf{1} \quad &\leq \quad \mathbf{0} \\
\tilde{\alpha} \quad &\geq \quad \mathbf{0}
\end{aligned}
\tag{1}
$$

where

$$
\tilde{\alpha} = \begin{bmatrix} \alpha \\ \alpha^* \end{bmatrix}, \quad
\bar{\mathbf{y}} = \begin{bmatrix} \mathbf{y} - \epsilon\mathbf{1} \\ -\mathbf{y} - \epsilon\mathbf{1} \end{bmatrix}, \quad
D = \begin{bmatrix} K & -K \\ -K & K \end{bmatrix}, \quad
\mathbf{c} = \begin{bmatrix} \mathbf{1} \\ -\mathbf{1} \end{bmatrix}
$$

The basic idea of decomposition is to split the variable vector $\tilde{\alpha}$ into the working set $\tilde{\alpha}_B$ of fixed size $q$ and the non-working set $\tilde{\alpha}_N$ containing the rest of the variables. The corresponding parts of vectors $\mathbf{c}$ and $\bar{\mathbf{y}}$ will also bear subscripts $N$ and $B$. The matrix $D$ is partitioned into $D_{BB}$, $D_{BN} = D_{NB}^T$ and $D_{NN}$. A further requirement is that, for the $i$-th element of the training data, both $\alpha_i$ and $\alpha_i^*$ are either included in or omitted from the working set.[1] The values of the variables in the non-working set are frozen for the iteration, and optimization is only performed with respect to the variables in the working set.

Optimization of the working set is also a quadratic program. This can be seen by re-arranging the terms of the objective function and the equality constraint in

(1) and dropping the terms independent of $\tilde{\alpha}_B$ from the objective. The resulting quadratic program (sub-problem) is formulated as follows:

$$\text{Maximize} \quad W_B(\tilde{\alpha}_B) \quad = \quad (\tilde{\mathbf{y}}_B^T - \tilde{\alpha}_N^T D_{NB})\tilde{\alpha}_B - \frac{1}{2}\tilde{\alpha}_B^T D_{BB}\tilde{\alpha}_B$$

$$\text{subject to:} \quad \begin{aligned} \mathbf{c}_B^T \tilde{\alpha}_B + \mathbf{c}_N^T \tilde{\alpha}_N &= 0 \\ \tilde{\alpha}_B - C\mathbf{1} &\leq 0 \\ \tilde{\alpha}_B &\geq 0 \end{aligned} \tag{2}$$

The basic decomposition algorithm chooses the first working set at random, and proceeds iteratively by selecting sub-optimal working sets and re-optimizing them, by solving quadratic program (2), until all subsets of size $q$ are optimal. The precise formulation of termination conditions will be developed in the following section.

## 3  Optimality of a Working Set

In order to maintain strict improvement of the objective function, the working set must be sub-optimal before re-optimization. The classical Karush-Kuhn-Tucker (KKT) conditions are necessary and sufficient for optimality of a quadratic program. I will use these conditions applied to the standard form of a quadratic program, as described in [5], p. 36.

The standard form of a quadratic program requires that all constraints are of equality type except for non-negativity constraints. To cast the regression SVM quadratic program (1) into the standard form, the slack variables $\mathbf{s}^T = (s_1, \ldots, s_{2l})$ corresponding to the box constraints, and the following matrices are introduced:

$$E = \begin{bmatrix} \mathbf{1} & I & \mathbf{0} \\ -\mathbf{1} & \mathbf{0} & I \end{bmatrix}, \quad \tilde{E} = \begin{bmatrix} E \\ I \end{bmatrix}, \quad \mathbf{z} = \begin{bmatrix} \tilde{\alpha} \\ 0 \\ \mathbf{s} \end{bmatrix}, \quad \mathbf{f} = \begin{bmatrix} 0 \\ \mathbf{C} \end{bmatrix} \tag{3}$$

where $\mathbf{1}$ is a vector of length $l$, $\mathbf{C}$ is a vector of length $2l$. The zero element in vector $\mathbf{z}$ reflects the fact that a slack variable for the equality constraint must be zero. In the matrix notation all constraints of problem (1) can be compactly expressed as:

$$\begin{aligned} \tilde{E}^T \mathbf{z} &= \mathbf{f} \\ \mathbf{z} &\geq \mathbf{0} \end{aligned} \tag{4}$$

In this notation the Karush-Kuhn-Tucker Theorem can be stated as follows:

**Theorem 1 (Karush-Kuhn-Tucker Theorem)** *The primal vector $\mathbf{z}$ solves the quadratic problem (1) if and only if it satisfies (4) and there exists a dual vector $\mathbf{u}^T = (\mathbf{\Pi}^T \; \mathbf{w}^T) = (\mathbf{\Pi}^T \; (\mu \; \mathbf{\Upsilon}^T))$ such that:*

$$\mathbf{\Pi} = D\tilde{\alpha} + E\mathbf{w} - \tilde{\mathbf{y}} \geq \mathbf{0} \tag{5}$$

$$\mathbf{\Upsilon} \geq \mathbf{0} \tag{6}$$

$$\mathbf{u}^T \mathbf{z} = 0 \tag{7}$$

It follows from the Karush-Kuhn-Tucker Theorem that if for all $\mathbf{u}$ satisfying conditions (6) – (7) the system of inequalities (5) is inconsistent then the solution of problem (1) is *not* optimal. Since the objective function of sub-problem (2) was obtained by merely re-arranging terms in the objective function of the initial problem (1), the same conditions guarantee that the sub-problem (2) is not optimal. Thus, the main strategy for identifying sub-optimal working sets will be to enforce inconsistency of the system (5) while satisfying conditions (6) – (7).

Let us further analyze inequalities in (5). Each inequality has one of the following forms:

$$\pi_i = -\phi_i + \epsilon + v_i + \mu \geq 0 \tag{8}$$

$$\pi_i^* = \phi_i + \epsilon - v_i^* - \mu \geq 0 \tag{9}$$

where

$$\phi_i = y_i - \sum_{j=1}^{l}(\alpha_j - \alpha_j^*)K_{ij}$$

Consider the values $\alpha_i$ can possible take:

1. $\alpha_i = 0$. In this case $s_i = C$, and, by complementarity condition (7), $v_i = 0$. Then inequality (8) becomes:

$$\pi_i = -\phi_i + \epsilon + \mu \geq 0 \quad \Rightarrow \quad \mu \geq \phi_i - \epsilon$$

2. $\alpha_i = C$. By complementarity condition (7), $\pi_i = 0$. Then inequality (8) becomes:

$$-\phi_i + \epsilon + \mu + v_i = 0 \quad \Rightarrow \quad \mu \leq \phi_i - \epsilon$$

3. $0 < \alpha_i < C$. By complementarity condition (7), $v_i = 0$, $\pi_i = 0$. Then inequality (8) becomes:

$$-\phi_i + \epsilon + \mu = 0 \quad \Rightarrow \quad \mu = \phi_i - \epsilon$$

Similar reasoning for $\alpha_i^*$ and inequality (9) yields the following results:

1. $\alpha_i^* = 0$. Then

$$\mu \leq \phi_i + \epsilon$$

2. $\alpha_i^* = C$. Then

$$\mu \geq \phi_i + \epsilon$$

3. $0 < \alpha_i^* < C$. Then

$$\mu = \phi_i + \epsilon$$

As one can see, the only free variable in system (5) is $\mu$. Each inequality restricts $\mu$ to a certain interval on a real line. Such intervals will be denoted as *$\mu$-sets* in the rest of the exposition. Any subset of inequalities in (5) is inconsistent if the intersection of the corresponding $\mu$-sets is empty. This provides a lucid rule for determining optimality of any working set: it is sub-optimal if the intersection of $\mu$-sets of all its points is empty. A sub-optimal working set will also be denoted as "inconsistent". The following summarizes the rules for calculation of $\mu$-sets, taking into account that for regression SVM $\alpha_i\alpha_i^* = 0$:

$$\mathcal{M}_i = \begin{cases} [\phi_i - \epsilon, \phi_i + \epsilon], & \text{if } \alpha_i = 0,\ \alpha_i^* = 0 \\ [\phi_i - \epsilon, \phi_i - \epsilon], & \text{if } 0 < \alpha_i < C,\ \alpha_i^* = 0 \\ (-\infty, \phi_i - \epsilon], & \text{if } \alpha_i = C,\ \alpha_i^* = 0 \\ [\phi_i + \epsilon, \phi_i + \epsilon], & \text{if } \alpha_i = 0,\ 0 < \alpha_i^* < C \\ [\phi_i + \epsilon, +\infty), & \text{if } \alpha_i = 0,\ \alpha_i^* = C \end{cases} \tag{10}$$

## 4   Maximal Inconsistency Algorithm

While inconsistency of the working set at each iteration guarantees convergence of decomposition, the rate of convergence is quite slow if arbitrary inconsistent working sets are chosen. A natural heuristic is to select "maximally inconsistent" working sets, in a hope that such choice would provide the greatest improvement of the objective function. The notion of "maximal inconsistency" is easy to define: let it be the gap between the smallest right boundary and the largest left boundary of $\mu$-sets of elements in the training set:

$$G = L - R$$
$$L = \max_{0 < i < l} \mu_i^l, \quad R = \min_{0 < i < l} \mu_i^r$$

where $\mu_i^l$, $\mu_i^r$ are the left and the right boundaries respectively (possibly minus or plus infinity) of the $\mu$-set $\mathcal{M}_i$. It is convenient to require that the largest possible inconsistency gap be maintained between all pairs of points comprising the working set. The obvious implementation of such strategy is to select $q/2$ elements with the largest values of $\mu^l$ and $q/2$ elements with the smallest values of $\mu^r$. The maximal inconsistency strategy is summarized in Algorithm 1.

---

**Algorithm 1** Maximal inconsistency SVM decomposition algorithm.

---

Let $S$ be the list of all samples.
**while** $(L > R)$

- compute $\mathcal{M}_i$ according to the rules (10) for all elements in $S$
- select $q/2$ elements with the largest values of $\mu^l$ ("left pass")
- select $q/2$ elements with the smallest values of $\mu^r$ ("right pass")
- re-optimize the working set

---

Although the motivation provided for the maximal inconsistency algorithm is purely heuristic, the algorithm can be rigorously derived, in a similar fashion as Joachims' algorithm, from Zoutendijk's feasible direction problem. Details of such derivation cannot be presented here due to space constraints. Because of this relationship I will further refer to both algorithms as "feasible direction" algorithms.

## 5   Experimental Results

Experimental evaluation of the new algorithm was performed on the modified KDD Cup 1998 data set. The original data set is available under *http://www.ics.uci.edu/~kdd/databases/kddcup98/kddcup98.html*. The following modifications were made to obtain a pure regression problem:

- All 75 character fields were eliminated.
- Numeric fields CONTROLN, ODATEDW, TCODE and DOB were elimitated.

The remaining 400 features and the labels were scaled between 0 and 1. Initial subsets of the training database of different sizes were selected for evaluation of the scaling properties of the new algorithm. The training times of the algorithms, with and without decomposition, the numbers of support vectors, including bounded support vectors, and the experimental scaling factors, are displayed in Table 1.

Table 1: Training time (sec) and number of SVs for the KDD Cup problem

| Examples | no dcmp | dcmp | total SV | BSV |
|---|---|---|---|---|
| 500 | 39 | 10 | 274 | 0 |
| 1000 | 226 | 41 | 518 | 3 |
| 2000 | 1490 | 158 | 970 | 5 |
| 3000 | 5744 | 397 | 1429 | 7 |
| 5000 | 27052 | 1252 | 2349 | 15 |
| scaling factor: | 2.84 | 2.08 | | |
| SV-scaling factor: | 3.06 | 2.24 | | |

Table 2: Training time (sec) and number of SVs for the KDD Cup problem, reduced feature space.

| Examples | no dcmp | dcmp | total SV | BSV |
|---|---|---|---|---|
| 500 | 56 | 18 | 170 | 30 |
| 1000 | 346 | 44 | 374 | 62 |
| 2000 | 1768 | 198 | 510 | 144 |
| 3000 | 4789 | 366 | 729 | 222 |
| 5000 | 22115 | 863 | 1139 | 354 |
| scaling factor: | 2.55 | 1.72 | | |
| SV-scaling factor: | 3.55 | 2.35 | | |

The experimental scaling factors are obtained by fitting lines to log-log plots of the running times against sample sizes, in the number of examples and the number of unbounded support vectors respectively. Experiments were run on SGI Octane with 195MHz clock and 256M RAM. RBF kernel with $\gamma = 10$, $C = 1$, termination accuracy 0.001, working set size of 20, and cache size of 5000 samples were used.

A similar experiment was performed on a reduced feature set consisting of the first 50 features selected from the full-size data set. This experiment illustrates the behavior of the algorithms when the large number of support vectors are bounded. The results are presented in Table 2.

# 6   Discussion

It comes at no surprise that the decomposition algorithm outperforms the conventional training algorithm by an order of magnitude. Similar results have been well established for pattern recognition SVM. Remarkable is the co-incidence of scaling factors of the maximal inconsistency algorithm and Joachims' algorithm: his scaling factors range from 1.7 to 2.1 [3]. I believe however, that a more important performance measure is SV-scaling factor, and the results above suggest that this factor is consistent even for problems with significantly different compositions of support vectors. Further experiments should investigate properties of this measure.

Finally, I would like to mention other methods proposed in order to speed-up training of SVM, although no experimental results have been reported for these methods with regard to training of the regression SVM. Chunking [6], p. 366, iterates through

the training data accumulating support vectors and adding a "chunk" of new data until no more changes to a solution occur. The main problem with this method is that when the percentage of support vectors is high it essentially solves the problem of almost the same size more than once. Sequential Minimal Optimization (SMO), proposed by Platt [7] and easily extendable to the regression SVM [1], employs an idea similar to decomposition but always uses the working set of size 2. For such a working set, a solution can be calculated "by hand" without numerical optimization. A number of heuristics is applied in order to choose a good working set. It is difficult to draw a comparison between the working set selection mechanisms of SMO and the feasible direction algorithms but experimental results of Joachims [3] suggest that SMO is slower. Another advantage of feasible direction algorithms is that the size of the working set is not limited to 2, as in SMO. Practical experience shows that the optimal size of the working set is between 10 and 100. Lastly, traditional optimization methods, such as Newton's or conjugate gradient methods, can be modified to yield the complexity of $O(s^3)$, where $s$ is the number of detected support vectors [8]. This can be a considerable improvement over the methods that have complexity of $O(l^3)$, where $l$ is the total number of training samples.

The real challenge lies in attaining sub-$O(s^3)$ complexity. While the experimental results suggest that feasible direction algorithms might attain such complexity, their complexity is not fully understood from the theoretical point of view. More specifically, the convergence rate, and its dependence on the number of support vectors, needs to be analyzed. This will be the main direction of the future research in feasible direction SVM training algorithms.

**References**

[1] Smola, A., Schölkopf, B. (1998) A Tutorial on Support Vector Regression. *NeuroCOLT2 Technical Report NC2-TR-1998-030.*

[2] Osuna, E., Freund, R., Girosi, F. (1997) An Improved Training Algorithm for Support Vector Machines. *Proceedings of IEEE NNSP'97.* Amelia Island FL.

[3] Joachims, T. (1998) Making Large-Scale SVM Learning Practical. *Advances in Kernel Methods – Support Vector Learning.* B. Schölkopf, C. Burges, A. Smola, (eds.) MIT-Press.

[4] Osuna, E. (1998) Support Vector Machines: Training and Applications. Ph. D. Dissertation. Operations Research Center, MIT.

[5] Boot, J. (1964) *Quadratic Programming. Algorithms – Anomalies – Applications.* North Holland Publishing Company, Amsterdam.

[6] Vapnik, V. (1982) *Estimation of Dependencies Based on Empirical Data.* Springer-Verlag.

[7] Platt, J. (1998) Fast Training of Support Vector Machines Using Sequential Minimal Optimization. *Advances in Kernel Methods – Support Vector Learning.* B. Schölkopf, C. Burges, A. Smola, (eds.) MIT-Press.

[8] Kaufman, L. (1998) Solving the Quadratic Programming Problem Arising in SupportVector Classification. *Advances in Kernel Methods – Support Vector Learning.* B. Schölkopf, C. Burges, A. Smola, (eds.) MIT-Press.

## Footnotes

[1] This rule facilitates formulation of sub-problems to be solved at each iteration.
